# A concave regularization technique for sparse mixture models

**Martin Larsson**
School of Operations Research and Information Engineering
Cornell University
mol23@cornell.edu

**Johan Ugander**
Center for Applied Mathematics
Cornell University
jhu5@cornell.edu

## Abstract

Latent variable mixture models are a powerful tool for exploring the structure in large datasets. A common challenge for interpreting such models is a desire to impose sparsity, the natural assumption that each data point only contains few latent features. Since mixture distributions are constrained in their $L_1$ norm, typical sparsity techniques based on $L_1$ regularization become toothless, and concave regularization becomes necessary. Unfortunately concave regularization typically results in EM algorithms that must perform problematic non-concave M-step maximizations. In this work, we introduce a technique for circumventing this difficulty, using the so-called Mountain Pass Theorem to provide easily verifiable conditions under which the M-step is well-behaved despite the lacking concavity. We also develop a correspondence between logarithmic regularization and what we term the pseudo-Dirichlet distribution, a generalization of the ordinary Dirichlet distribution well-suited for inducing sparsity. We demonstrate our approach on a text corpus, inferring a sparse topic mixture model for 2,406 weblogs.

## 1 Introduction

The current trend towards 'big data' has created a strong demand for techniques to efficiently extract structure from ever-accumulating unstructured datasets. Specific contexts for this demand include latent semantic models for organizing text corpora, image feature extraction models for navigating large photo datasets, and community detection in social networks for optimizing content delivery. Mixture models identify such latent structure, helping to categorize unstructured data.

Mixture models approach datasets as a set $\mathcal{D}$ of element $d \in \mathcal{D}$, for example images or text documents. Each element consists of a collection of words $w \in \mathcal{W}$ drawn with replacement from a vocabulary $\mathcal{W}$. Each element-word pair observation is further assumed to be associated with an unobserved class $z \in \mathcal{Z}$, where $\mathcal{Z}$ is the set of classes. Ordinarily it is assumed that $|\mathcal{Z}| \ll |\mathcal{D}|$, namely that the number of classes is much less than the number of elements. In this work we explore an additional sparsity assumption, namely that individual elements only incorporate a small subset of the $|\mathcal{Z}|$ classes, so that each element arises as a mixture of only $\ell \ll |\mathcal{Z}|$ classes. We develop a framework to overcome mathematical difficulties in how this assumption can be harnessed to improve the performance of mixture models.

Our primary context for mixture modeling in this work will be latent semantic models of text data, where elements $d$ are documents, words $w$ are literal words, and classes $z$ are vocabulary topics. We apply our framework to models based on Probabilistic Latent Semantic Analysis (PLSA) [1]. While PLSA is often outperformed within text applications by techniques such as Latent Dirichlet Allocation (LDA) [2], it forms the foundation of many mixture model techniques, from computer vision [3] to network community detection [4], and we emphasize that our contribution is an optimization technique intended for broad application outside merely topic models for text corpora. The

near-equivalence between PLSA and Nonnegative Matrix Factorization (NMF) [5, 6] implies that our technique is equally applicable to NMF problems as well. Sparse inference as a rule targets point estimation, which makes PLSA-style models appropriate since they are inherently frequentist, deriving point-estimated models via likelihood maximization. In contrast, fully Bayesian frameworks such as Latent Dirichlet Allocation (LDA) output a posterior distribution across the model space.

Sparse inference is commonly achieved through two largely equivalent techniques: regularization or MAP inference. Regularization modifies ordinary likelihood maximization with a penalty on the magnitudes of the parameters. Maximum a posteriori (MAP) inference employs priors concentrated towards small parameter values. MAP PLSA is an established technique [7], but earlier work has been limited to log-concave prior distributions (corresponding to convex regularization functions) that make a concave contribution to the posterior log-likelihood. While such priors allow for tractable EM algorithms, they have the effect of promoting smoothing rather than sparsity. In contrast, sparsity-inducing priors are invariably convex in their contribution. In this work we resolve this difficulty by showing how, even though concavity fails in general, we are able to derive simple checkable conditions that guarantee a unique stationary point to the M-step objective function that serves as the unique global maximum. This rather surprising result, using the so-called Mountain Pass Theorem, is a noteworthy contribution to the theory of learning algorithms which we expect has many applications outside merely PLSA.

Section 2 briefly outlines the structure of MAP inference for PLSA. Section 3 discusses priors appropriate for inducing sparsity, and introduces a generalization of the Dirichlet distribution which we term the pseudo-Dirichlet distribution. Section 4 contains our main result, a tractable EM algorithm for PLSA under sparse pseudo-Dirichlet priors using the Mountain Pass Theorem. Section 5 presents empirical results for a corpus of 2,406 weblogs, and section 6 concludes with a discussion.

## 2 Background and preliminaries

### 2.1 Standard PLSA

Within the PLSA framework, word-document-topic triplets $(w, d, z)$ are assumed to be i.i.d. draws from a joint distribution on $\mathcal{W} \times \mathcal{D} \times \mathcal{Z}$ of the form

$$P(w, d, z \mid \theta) = P(w \mid z)P(z \mid d)P(d), \tag{1}$$

where $\theta$ consists of the model parameters $P(w \mid z)$, $P(z \mid d)$ and $P(d)$ for $(w, d, z)$ ranging over $\mathcal{W} \times \mathcal{D} \times \mathcal{Z}$. Following [1], the corresponding data log-likelihood can be written

$$\ell_0(\theta) = \sum_{w,d} n(w, d) \log \left[ \sum_z P(w \mid z)P(z \mid d) \right] + \sum_d n(d) \log P(d), \tag{2}$$

where $n(w, d)$ is the number of occurrences of word $w$ in document $d$, and $n(d) = \sum_w n(w, d)$ is the total number of words in $d$. The goal is to maximize the likelihood over the set of admissible $\theta$. This is accomplished using the EM algorithm, iterating between the following two steps:

**E-step:** Find $P(z \mid w, d, \theta')$, the posterior distribution of the latent variable $z$, given $(w, d)$ and a current parameter estimate $\theta'$.

**M-step:** Maximize $Q_0(\theta \mid \theta')$ over $\theta$, where

$$Q_0(\theta \mid \theta') = \sum_d n(d) \log P(d) + \sum_{w,d,z} n(w, d)P(z \mid w, d, \theta') \log \left[ P(w \mid z)P(z \mid d) \right].$$

We refer to [1] for details on the derivations, as well as extensions using so-called tempered EM. The resulting updates corresponding to the E-step and M-step are, respectively,

$$P(z \mid w, d, \theta) = \frac{P(w \mid z)P(z \mid d)}{\sum_{z'} P(w \mid z')P(z' \mid d)} \tag{3}$$

and

$$P(w \mid z) = \frac{\sum_d P(z \mid w, d, \theta')n(w, d)}{\sum_{w',d} P(z \mid w', d, \theta')n(w', d)},$$

$$P(z \mid d) = \frac{\sum_w P(z \mid w, d, \theta')n(w, d)}{n(d)}, \qquad P(d) = \frac{n(d)}{\sum_{d'} n(d')}. \tag{4}$$

Note that PLSA has an alternative parameterization, where (1) is replaced by $P(w, d, z \mid \theta) = P(w|z)P(d|z)P(z)$. This formulation is less interesting in our context, since our sparsity assumption is intended as a statement about the vectors $(P(z \mid d) : z \in \mathcal{Z}), d \in \mathcal{D}$.

## 2.2 MAP PLSA

The standard MAP extension of PLSA is to introduce a prior density $P(\theta)$ on the parameter vector, and then maximize the posterior data log-likelihood $\ell(\theta) = \ell_0(\theta) + \log P(\theta)$ via the EM algorithm.

In order to simplify the optimization problem, we impose the reasonable restriction that the vectors $(P(w \mid z) : w \in \mathcal{W})$ for $z \in \mathcal{Z}$, $(P(z \mid d) : z \in \mathcal{Z})$ for $d \in \mathcal{D}$, and $(P(d) : d \in \mathcal{D})$ be mutually independent under the prior $P(\theta)$. That is,

$$P(\theta) = \prod_{z \in \mathcal{Z}} f_z(P(w \mid z) : w \in \mathcal{W}) \times \prod_{d \in \mathcal{D}} g_d(P(z \mid d) : z \in \mathcal{Z}) \times h(P(d) : d \in \mathcal{D}),$$

for densities $f_z$, $g_d$ and $h$ on the simplexes in $\mathbb{R}^{|\mathcal{W}|}$, $\mathbb{R}^{|\mathcal{Z}|}$ and $\mathbb{R}^{|\mathcal{D}|}$, respectively. With this structure on $P(\theta)$ one readily verifies that the M-step objective function for the MAP likelihood problem, $Q(\theta \mid \theta') = Q_0(\theta \mid \theta') + \log P(\theta)$, is given by

$$Q(\theta \mid \theta') = \sum_z F_z(\theta \mid \theta') + \sum_d G_d(\theta \mid \theta') + H(\theta \mid \theta'),$$

where

$$F_z(\theta \mid \theta') = \sum_{w,d} P(z \mid w, d, \theta')n(w,d) \log P(w \mid z) + \log f_z(P(w \mid z) : w \in \mathcal{W}),$$

$$G_d(\theta \mid \theta') = \sum_{w,z} P(z \mid w, d, \theta')n(w,d) \log P(z \mid d) + \log g_d(P(z \mid d) : z \in \mathcal{Z}),$$

$$H(\theta \mid \theta') = \sum_d n(d) \log P(d) + \log h(P(d) : d \in \mathcal{D}).$$

As a comment, notice that if the densities $f_z$, $g_d$, or $h$ are log-concave then $F_z$, $G_d$, and $H$ are concave in $\theta$. Furthermore, the functions $F_z$, $G_d$, and $H$ can be maximized independently, since the corresponding non-negativity and normalization constraints are decoupled. In particular, the $|\mathcal{Z}| + |\mathcal{D}| + 1$ optimization problems can be solved in parallel.

# 3 The pseudo-Dirichlet prior

The parameters for PLSA models consist of $|\mathcal{Z}| + |\mathcal{D}| + 1$ probability distributions taking their values on $|\mathcal{Z}| + |\mathcal{D}| + 1$ simplexes. The most well-known family of distributions on the simplex is the Dirichlet family, which has many properties that make it useful in Bayesian statistics [8]. Unfortunately the Dirichlet distribution is not a suitable prior for modeling sparsity for PLSA, as we shall see, and to address this we introduce a generalization of the Dirichlet distribution which we call the pseudo-Dirichlet distribution.

To illustrate why the Dirichlet distribution is unsuitable in the present context, consider placing a symmetric Dirichlet prior on $(P(z \mid d) : z \in \mathcal{Z})$ for each document $d$. That is, for each $d \in \mathcal{D}$,

$$g_d(P(z \mid d) : z \in \mathcal{Z}) \propto \prod_{z \in \mathcal{Z}} P(z \mid d)^{\alpha - 1},$$

where $\alpha > 0$ is the concentration parameter. Let $f_z$ and $h$ be constant. The relevant case for sparsity is when $\alpha < 1$, which concentrates the density toward the (relative) boundary of the simplex. It is easy to see that the distribution is in this case log-convex, which means that the contribution to the log-likelihood and M-step objective function $G_d(\theta \mid \theta')$ will be convex. We address this problem in Section 4. A bigger problem, however, is that for $\alpha < 1$ the density of the symmetric Dirichlet distribution is unbounded and the MAP likelihood problem does not have a well-defined solution, as the following result shows.

**Proposition 1** *Under the above assumptions on $f_z$, $g_d$ and $h$ there are infinitely many sequences $(\theta_m)_{m \geq 1}$, converging to distinct limits, such that $\lim_{m \to \infty} Q(\theta_m \mid \theta_m) = \infty$. As a consequence, $\ell(\theta_m)$ tends to infinity as well.*

**Proof.** Choose $\theta_m$ as follows: $P(d) = |\mathcal{D}|^{-1}$ and $P(w \mid z) = |\mathcal{W}|^{-1}$ for all $w, d$ and $z$. Fix $d_0 \in \mathcal{D}$ and $z_0 \in \mathcal{Z}$, and set $P(z_0 \mid d_0) = m^{-1}$, $P(z \mid d_0) = \frac{1 - m^{-1}}{|\mathcal{Z}| - 1}$ for $z \neq z_0$, and $P(z \mid d) = |\mathcal{Z}|^{-1}$ for all $z$ and $d \neq d_0$. It is then straightforward to verify that $Q(\theta_m \mid \theta_m)$ tends to infinity. The choice of $d_0$ and $z_0$ was arbitrary, so by choosing two other points we get a different sequence with a different limit. Taking convex combinations yields the claimed infinity of sequences. The second statement follows from the well-known fact that $Q(\theta \mid \theta') \leq \ell(\theta)$ for all $\theta$ and $\theta'$. $\square$

This proposition is a formal statement of the observation that when the Dirichlet prior is unbounded, any single zero element in $P(z|d)$ leads to an infinite posterior likelihood, and so the optimization problem is not well-posed. To overcome these unbounded Dirichlet priors while retaining their sparsity-inducing properties, we introduce the following class of distributions on the simplex.

**Definition 1** *A random vector confined to the simplex in $\mathbb{R}^p$ is said to follow a* pseudo-Dirichlet *distribution with concentration parameter $\boldsymbol{\alpha} = (\alpha_1, \ldots, \alpha_p) \in \mathbb{R}^p$ and perturbation parameter $\boldsymbol{\epsilon} = (\epsilon_1, \ldots, \epsilon_p) \in \mathbb{R}_+^p$ if it has a density on the simplex given by*

$$P(x_1, \ldots, x_p \mid \boldsymbol{\alpha}, \boldsymbol{\epsilon}) = C \prod_{i=1}^{p} (\epsilon_i + x_i)^{\alpha_i - 1} \tag{5}$$

*for a normalizing constant $C$ depending on $\boldsymbol{\alpha}$ and $\boldsymbol{\epsilon}$. If $\alpha_i = \alpha$ and $\epsilon_i = \epsilon$ for all $i$ and some fixed $\alpha \in \mathbb{R}$, $\epsilon \geq 0$, we call the resulting distribution* symmetric pseudo-Dirichlet.

Notice that if $\epsilon_i > 0$ for all $i$, the pseudo-Dirichlet density is indeed bounded for all $\boldsymbol{\alpha}$. If $\epsilon_i = 0$ and $\alpha_i > 0$ for all $i$, we recover the standard Dirichlet distribution. If $\epsilon_i = 0$ and $\alpha_i \leq 0$ for some $i$ then the density is not integrable, but can still be used as an improper prior. Like the Dirichlet distribution, when $\alpha < 1$ the pseudo-Dirichlet distribution is log-convex, and it will make a convex contribution to the M-step objective function of any EM algorithm.

The psuedo-Dirichlet distribution can be viewed as a bounded perturbation of the Dirichlet distribution, and for small values of the perturbation parameter $\boldsymbol{\epsilon}$, many of the properties of the original Dirichlet distribution hold approximately. In our discussion section we offer a justification for allowing $\alpha \leq 0$, framed within a regularization approach.

## 4 EM under pseudo-Dirichlet priors

We now derive an EM algorithm for PLSA under sparse pseudo-Dirichlet priors. The E-step is the same as for standard PLSA, and is given by (3). The M-step consists in optimizing each $F_z$, $G_d$ and $H$ individually. While our M-step will not offer a closed-form maximization, we are able to derive simple checkable conditions under which the M-step has a stationary point that is also the global maximum. Once the conditions are satisfied, the M-step optimum can be found via a practitioner's favorite root-finding algorithm. For consideration, we propose an iteration scheme that in practice we find converges rapidly and well. Because our sparsity assumption focuses on the parameters $P(z|d)$, we perform our main analysis on $G_d$, but for completeness we state the corresponding result for $F_z$. The less applicable treatment of $H$ is omitted.

Consider the problem of maximizing $G_d(\theta \mid \theta')$ over $(P(z \mid d) : z \in \mathcal{Z})$ subject to $\sum_z P(z \mid d) = 1$ and $P(z \mid d) \geq 0$ for all $z$. We use symmetric pseudo-Dirichlet priors with parameters $\boldsymbol{\alpha}_d = (\alpha_d, \ldots, \alpha_d)$ and $\boldsymbol{\epsilon}_d = (\epsilon_d, \ldots, \epsilon_d)$ for $\alpha_d \in \mathbb{R}$ and $\epsilon_d > 0$. Since each $G_d$ is treated separately, let us fix $d$ and write

$$x_z = P(z \mid d), \qquad c_z = \sum_w P(z \mid w, d, \theta') n(w, d),$$

where the dependence on $d$ is suppressed in the notation. For $\boldsymbol{x} = (x_z : z \in \mathcal{Z})$ and a fixed $\theta'$, we write $G_d(\boldsymbol{x}) = G_d(\theta \mid \theta')$, which yields, up to an additive constant,

$$G_d(\boldsymbol{x}) = \sum_z \left[ (\alpha_d - 1) \log(\epsilon_d + x_z) + c_z \log x_z \right].$$

The task is to maximize $G_d$, subject to $\sum_z x_z = 1$ and $x_z \geq 0$ for all $z$. Assuming that every word $w$ is observed in at least one document $d$ and that all components of $\theta'$ are strictly positive, Lemma 1 below implies that any M-step optimizer must have strictly positive components. The non-negativity constraint is therefore never binding, so the appropriate Lagrangian for this problem is

$$\mathcal{L}_d(\boldsymbol{x}; \lambda) = G_d(\boldsymbol{x}) + \lambda \Big[ 1 - \sum_z x_z \Big],$$

and it suffices to consider its stationary points.

**Lemma 1** *Assume that every word $w$ has been observed in at least one document $d$, and that $P(z \mid w, d; \theta') > 0$ for all $(w, d, z)$. If $x_z \to 0$ for some $z$, and the nonnegativity and normalization constraints are maintained, then $G_d(\boldsymbol{x}) \to -\infty$.*

**Proof.** The assumption implies that $c_z > 0$, $\forall z$. Therefore, since $\log(\epsilon_d + x_z)$ and $\log x_z$ are bounded from above, $\forall z$, when $\theta$ stays in the feasible region, $x_z \to 0$ leads to $G_d(\boldsymbol{x}) \to -\infty$. $\square$

The next lemma establishes a property of the stationary points of the Lagrangian $\mathcal{L}_d$ which will be the key to proving our main result.

**Lemma 2** *Let $(\boldsymbol{x}, \lambda)$ be any stationary point of $\mathcal{L}_d$ such that $x_z > 0$ for all $z$. Then $\lambda \geq n(d) - (1 - \alpha_d)|\mathcal{Z}|$. If in addition to the assumptions of Lemma 1 we have $n(d) \geq (1 - \alpha_d)|\mathcal{Z}|$, then*

$$\frac{\partial^2 G_d}{\partial x_z^2}(\boldsymbol{x}, \lambda) < 0 \qquad \text{for all } z \in \mathcal{Z}.$$

**Proof.** We have $\frac{\partial \mathcal{L}_d}{\partial x_z} = \frac{c_z}{x_z} - \frac{1 - \alpha_d}{\epsilon_d + x_z} - \lambda$. Since $\frac{\partial \mathcal{L}_d}{\partial x_z}(\boldsymbol{x}, \lambda) = 0$ at the stationary point, we get $\lambda x_z = c_z - (1 - \alpha_d)\frac{x_z}{\epsilon_d + x_z} \geq c_z - (1 - \alpha_d)$, which, after summing over $z$ and using that $\sum_z x_z = 1$, yields

$$\lambda \geq \sum_z c_z - (1 - \alpha_d)|\mathcal{Z}|.$$

Furthermore, $\sum_z c_z = \sum_w n(w, d) \sum_z P(z \mid w, d, \theta') = n(d)$, so $\lambda \geq n(d) - (1 - \alpha_d)|\mathcal{Z}|$.

For the second assertion, using once again that $\frac{\partial \mathcal{L}_d}{\partial x_z}(\boldsymbol{x}, \lambda) = 0$ at the stationary point, a calculation shows that

$$\frac{\partial^2 G_d}{\partial x_z^2}(\boldsymbol{x}, \lambda) = -\frac{1}{x_z^2(\epsilon_d + x_z)}\Big[ x_z^2 \lambda + c_z \epsilon_d \Big].$$

The assumptions imply that $c_z > 0$, so it suffices to prove that $\lambda \geq 0$. This follows from our hypothesis and the first part of the lemma. $\square$

This allows us to obtain our main result result concerning the structure of the optimization problem associated with the M-step.

**Theorem 1** *Assume that (i) every word $w$ has been observed in at least one document $d$, (ii) $P(z \mid w, d, \theta') > 0$ for all $(w, d, z)$, and (iii) $n(d) > (1 - \alpha_d)|\mathcal{Z}|$ for each $d$. Then each Lagrangian $\mathcal{L}_d$ has a unique stationary point, which is the global maximum of the corresponding optimization problem, and whose components are strictly positive.*

The proof relies on the following version of the so-called Mountain Pass Theorem.

**Lemma 3 (Mountain Pass Theorem)** *Let $\mathcal{O} \subset \mathbb{R}^n$ be open, and consider a continuously differentiable function $\phi : \mathcal{O} \to \mathbb{R}$ s.t. $\phi(x) \to -\infty$ whenever $x$ tends to the boundary of $\mathcal{O}$. If $\phi$ has two distinct strict local maxima, it must have a third stationary point that is not a strict local maximum.*

**Proof.** See p. 223 in [9], or Theorem 5.2 in [10]. $\square$

**Proof of Theorem 1.** Consider a fixed $d$. We first prove that the corresponding Lagrangian $\mathcal{L}_d$ can have at most one stationary point. To simplify notation, assume without loss of generality that $\mathcal{Z} = \{1, \dots, K\}$, and define

$$\widetilde{G}_d(x_1, \dots, x_{K-1}) = G_d\Big( x_1, \dots, x_{K-1}, 1 - \sum_{k=1}^{K-1} x_k \Big).$$

The constrained maximization of $G_d$ is then equivalent to maximizing $\widetilde{G}_d$ over the open set $\mathcal{O} = \{(x_1, \ldots, x_{K-1}) \in \mathbb{R}_{++}^{K-1} : \sum_k x_k < 1\}$. The following facts are readily verified:

(i) If $(\boldsymbol{x}, \lambda)$ is a stationary point of $\mathcal{L}_d$, then $(x_1, \ldots, x_{K-1})$ is a stationary point of $\widetilde{G}_d$.

(ii) If $(x_1, \ldots, x_{K-1})$ is a stationary point of $\widetilde{G}_d$, then $(\boldsymbol{x}, \lambda)$ is a stationary point of $\mathcal{L}_d$, where $x_K = 1 - \sum_{k=1}^{K-1} x_k$ and $\lambda = \frac{c_K}{x_K} - \frac{1-\alpha_d}{\epsilon_d + x_K}$.

(iii) For any $\boldsymbol{y} = (y_1, \ldots, y_{K-1}, \sum_{k=1}^{K-1} y_k)$, we have $\boldsymbol{y}^T \nabla^2 \widetilde{G}_d \boldsymbol{y} = \sum_{k=1}^{K} y_k^2 \frac{\partial^2 G_d}{\partial x_k^2}$.

Now, suppose that $(\boldsymbol{x}, \lambda)$ is a stationary point of $\mathcal{L}_d$. Property (i) and property (iii) in conjunction with Lemma 2 imply that $(x_1, \ldots, x_{K-1})$ is a stationary point of $\widetilde{G}_d$ and that $\nabla^2 \widetilde{G}_d$ is negative definite there. Hence it is a strict local maximum.

Next, suppose for contradiction that there are two distinct such points. By Lemma 1, $\widetilde{G}_d$ tends to $-\infty$ near the boundary of $\mathcal{O}$, so we may apply the mountain pass theorem to get the existence of a third point $(\tilde{x}_1, \ldots, \tilde{x}_{K-1})$, stationary for $\widetilde{G}_d$, that is not a strict local maximum. But by (ii), this yields a corresponding stationary point $(\tilde{\boldsymbol{x}}, \tilde{\lambda})$ for $\mathcal{L}_d$. The same reasoning as above then shows that $(\tilde{x}_1, \ldots, \tilde{x}_{K-1})$ has to be a strict local max for $\widetilde{G}_d$, which is a contradiction. We deduce that $\mathcal{L}_d$ has at most one stationary point.

Finally, the continuity of $G_d$ together with its boundary behavior (Lemma 1) implies that a maximizer exists and has strictly positive components. But the maximizer must be a stationary point of $\mathcal{L}_d$, so together with the previously established uniqueness, the result follows. $\square$

Condition $(i)$ in Theorem 1 is not a real restriction, since a word that does not appear in any document typically will be removed from the vocabulary. Moreover, if the EM algorithm is initialized such that $P(z \mid w, d; \theta') > 0$ for all $(w, d, z)$, Theorem 1 ensures that this will be the case for all future iterates as well. The critical assumption is Condition $(iii)$. It can be thought of as ensuring that the prior does not drown the data. Indeed, sufficiently large negative values of $\alpha_d$, corresponding to strong prior beliefs, will cause the condition to fail.

While there are various methods available for finding the stationary point of $\mathcal{L}_d$, we have found that the following fixed-point type iterative scheme produces satisfactory results.

$$ x_z \leftarrow \frac{c_z}{n(d) + (1 - \alpha_d)\left[\frac{1}{\epsilon_d + x_z} - \sum_{z'} \frac{x_{z'}}{\epsilon_d + x_{z'}}\right]}, \qquad x_z \leftarrow \frac{x_z}{\sum_z x_z}. \tag{6} $$

To motivate this particular update rule, recall that

$$ \frac{\partial \mathcal{L}_d}{\partial x_z} = \frac{c_z}{x_z} - \frac{1 - \alpha_d}{\epsilon_d + x_z} - \lambda, \qquad \frac{\partial \mathcal{L}_d}{\partial \lambda} = 1 - \sum_z x_z. $$

At the stationary point, $\lambda x_z = c_z - \frac{1-\alpha_d}{\epsilon_d + x_z} x_z$, so by summing over $z$ and using that $\sum_z x_z = 1$ and $\sum_z c_z = n(d)$, we get $\lambda = n(d) - (1 - \alpha_d) \sum_z \frac{x_z}{\epsilon_d + x_z}$. Substituting this for $\lambda$ in $\frac{\partial \mathcal{L}_d}{\partial x_z} = 0$ and rearranging terms yields the first part of (6). Notice that Lemma 2 ensures that the denominator stays strictly positive. Further, the normalization is a classic technique to restrict $\boldsymbol{x}$ to the simplex. Note that (6) reduces to the standard PLSA update (4) if $\alpha_d = 1$.

For completeness we also consider the topic-vocabulary distribution $(P(w|z) : w \in \mathcal{W})$. We impose a symmetric pseudo-Dirichlet prior on the vector $(P(w \mid z) : w \in \mathcal{W})$ for each $z \in \mathcal{Z}$. The corresponding parameters are denoted by $\alpha_z$ and $\epsilon_z$. Each $F_z$ is optimized individually, so we fix $z \in \mathcal{Z}$ and write $y_w = P(w \mid z)$. The objective function $F_z(\boldsymbol{y}) = F_z(\theta \mid \theta')$ is then given by

$$ F_z(\boldsymbol{y}) = \sum_w \left[ (\alpha_z - 1) \log(\epsilon_z + y_w) + b_w \log y_z \right], \qquad b_w = \sum_d P(z \mid w, d, \theta') n(w, d). \tag{7} $$

The following is an analog of Theorem 1, whose proof is essentially the same and therefore omitted.

**Theorem 2** *Assume condition (i) and (ii) of Theorem 1 are satisfied, and that for each $z \in \mathcal{Z}$, $\sum_w b_w \geq (1 - \alpha_z)|\mathcal{W}|$. Then each $F_z$ has a unique local optimum on the simplex, which is also a global maximum and whose components are strictly positive.*

Unfortunately there is no simple expression for $\sum_w b_w$ in terms of the inputs to the problem. On the other hand, the sum can be evaluated at the beginning of each M-step, which makes it possible to verify that $\alpha_z$ is not too negative.

## 5 Empirical evaluation

To evaluate our framework for sparse mixture model inference, we develop a MAP PLSA topic model for a corpus of 2,406 blogger.com blogs, a dataset originally analyzed by Schler et al. [11] for the role of gender in language. Unigram frequencies for the blogs were built using the python NLTK toolkit [12]. Inference was run on the document-word distribution of 2,406 blogs and 2,000 most common words, as determined by the aggregate frequencies across the entire corpus. The implications of Section 4 is that in order to adapt PLSA for sparse MAP inference, we simply need to replace equation (4) from PLSA's ordinary M-step with an iteration of (6).

The corpus also contains a user-provided 'category' for each blog, indicating one of 28 categories. We focused our analysis on 8 varied but representative topics, while the complete corpus contained over 19,000 blogs. The user-provided topic labels are quite noisy, and so in order to have cleaner ground truth data for evaluating our model we chose to also construct a synthetic, sparse dataset. This synthetic dataset is employed to evaluate parameter choices within the model.

To generate our synthetic data, we ran PLSA on our text corpus and extracted the inferred $P(w|z)$ and $P(d)$ distributions, while creating 2,406 synthetic $P(z|d)$ distributions where each synthetic blog was a uniform mixture of between 1 and 4 topics. These distributions were then used to construct a ground-truth word-document distribution $Q(w, d)$, which we then sampled $N$ times, where $N$ is the total number of words in our true corpus. In this way we were able to generate a realistic synthetic dataset with a sparse and known document-topic distribution.

We evaluate the quality of each model by calculating the model perplexity of the reconstructed word-document distribution as compared to the underlying ground truth distribution used to generate the synthetic data. Here model perplexity is given by

$$\mathcal{P}(P(w,d)) = 2^{-\sum_{w,d} Q(w,d) \log_2 P(w,d)},$$

where $Q(w, d)$ is the true document-word distribution used to generate the synthetic dataset and $P(w, d)$ is the reconstructed matrix inferred by the model. Using this synthetic dataset we are able to evaluate the roles of $\alpha$ and $\epsilon$ in our algorithm, as seen in Figure 1.

From Figure 1 we can conclude that $\alpha$ should in practice be chosen close the algorithm's feasible lower bound, and $\epsilon$ can be almost arbitrarily small. Choosing $\alpha = \lceil 1 - \max_d n(d)/k \rceil$ and $\epsilon = 10^{-6}$, we return to our blog data with its user-provided labels. In Figure 2 we see that sparse inference indeed results in $P(z|d)$ distributions with significantly sparser support. Furthermore, we can more easily see how certain categories of blogs cluster in their usage of certain topics. For example, a majority of the blogs self-categorized as pertaining to 'religion' employ almost exclusively the second topic vocabulary of the model. The five most exceptional unigrams for this topic are 'prayer', 'christ', 'jesus', 'god', and 'church'.

## 6 Discussion

We have shown how certain latent variable mixture models can be tractably extended with sparsity-inducing priors using what we call the pseudo-Dirichlet distribution. Our main theoretical result shows that the resulting M-step maximization problem is well-behaved despite the lack of concavity, and empirical findings indicate that the approach is indeed effective. Our use of the Mountain Pass Theorem to prove that all local optima coincide is to the best of our knowledge new in the literature, and we find it intriguing and surprising that the global properties of maximizers, which are very rarely susceptible to analysis in the absence of concavity, can be studied using this tool.

The use of log-convex priors (equivalently, concave regularization functions) to encourage sparsity is particularly relevant when the parameters of the model correspond to probability distributions. Since each distribution has a fixed $L_1$ norm equal to one, the use of $L_1$-regularization, which otherwise would be the natural choice for inducing sparsity, becomes toothless. The pseudo-Dirichlet prior we introduce corresponds to a concave regularization of the form $\sum_i \log(x_i + \epsilon)$. We mention in

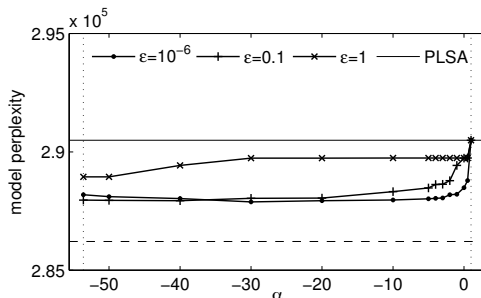

Figure 1: Model perplexity for inferred models with $k = 8$ topics as a function of the concentration parameter $\alpha$ of the pseudo-Dirichlet prior, shown from the algorithm's lower bound $\alpha = 1 - n(d)/k$ to the uniform prior case of $\alpha = 1$. Three different choices of $\epsilon$ are shown, as well as the baseline PLSA perplexity corresponding to a uniform prior. The dashed line indicates the perplexity $\mathcal{P}(Q(w, d))$, which should be interpreted as a lower bound.

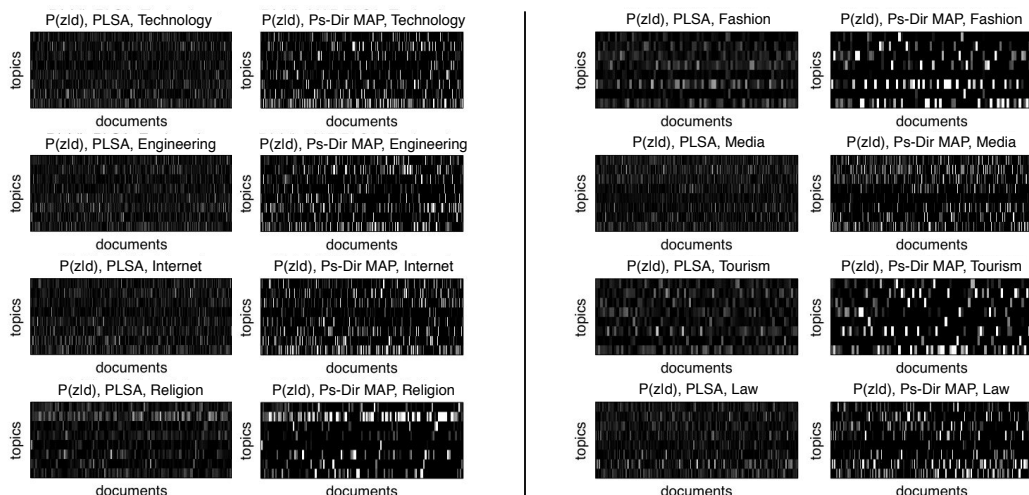

Figure 2: Document-topic distributions $P(z|d)$ for the 8 different categories of blogs studied. All distributions share the same color scale.

passing that the same sum-log regularization has also been used for sparse signal recovery in [13]. It should be emphasized that the notion of sparsity we discuss in this work is *not* in the formal sense of a small $L_0$ norm. Indeed, Theorem 1 shows that, no different from ordinary PLSA, the estimated parameters for MAP PLSA will all be strictly positive. Instead, we seek sparsity in the sense that most parameters should be almost zero.

Next, let us comment on the possibility to allow the concentration parameter $\alpha_d$ to be negative, assuming for simplicity that $f_z$ and $h$ are constant. Consider the normalized likelihood, where clearly $\ell(\theta)$ may be replaced by $\ell(\theta)/N$,

$$\frac{\ell(\theta)}{N} = \frac{\ell_0(\theta)}{N} - \sum_d \frac{1 - \alpha_d}{N} \sum_z \log(\epsilon_d + P(z \mid d)),$$

which by (2) we deduce only depends on the data through the normalized quantities $n(w, d)/N$. This indicates that the quantity $(1 - \alpha_d)/N$, which plays the role of a regularization 'gain' in the normalized problem, must be non-negligible in order for the regularization to have an effect. For realistic sizes of $N$, allowing $\alpha_d < 0$ therefore becomes crucial.

Finally, while we have chosen to present our methodology as applied to topic models, we expect the same techniques to be useful in a notably broader context. In particular, our methodology is directly applicable to problems solved through Nonnegative Matrix Factorization (NMF), a close relative of PLSA where matrix columns or rows are often similarly constrained in their $L_1$ norm.

**Acknowledgments:** This work is supported in part by NSF grant IIS-0910664.

# References

[1] T. Hofmann. Unsupervised learning by probabilistic latent semantic analysis. *Machine Learning*, 42:177–196, 2001.

[2] D.M. Blei, A.Y. Ng, and M.I. Jordan. Latent Dirichlet allocation. *Journal of Machine Learning Research*, 3:993–1022, 2003.

[3] A. Bosch, A. Zisserman, and X. Munoz. Scene Classification via pLSA. In *European Conference on Computer Vision*, 2006.

[4] I. Psorakis and B. Sheldon. Soft Partitioning in Networks via Baysian Non-negative Matrix Factorization. In *NIPS*, 2010.

[5] C. Ding, T. Li, and W. Peng. Nonnegative matrix factorization and probabilistic latent semantic indexing: Equivalence chi-square statistic, and a hybrid method. In *Proceedings of AAAI '06*, volume 21, page 342, 2006.

[6] E. Gaussier and C. Goutte. Relation between PLSA and NMF and implications. In *Proceedings of ACM SIGIR*, pages 601–602. ACM, 2005.

[7] A. Asuncion, M. Welling, P. Smyth, and Y.W. Teh. On smoothing and inference for topic models. In *Proc. of the 25th Conference on Uncertainty in Artificial Intelligence*, pages 27–34, 2009.

[8] A. Gelman. *Bayesian data analysis*. CRC Press, 2004.

[9] R. Courant. *Dirichlet's principle, conformal mapping, and minimal surfaces*. Interscience, New York, 1950.

[10] Y. Jabri. *The Mountain Pass Theorem: Variants, Generalizations and Some Applications*. Cambridge University Press, 2003.

[11] J. Schler, M. Koppel, S. Argamon, and J. Pennebaker. Effects of age and gender on blogging. In *Proc. of the AAAI Spring Symposium on Computational Approaches for Analyzing Weblogs*, pages 191–197, 2006.

[12] S. Bird, E. Klein, and Loper E. *Natural language processing with Python*. O'Reilly Media, 2009.

[13] E.J. Candès, M.B. Wakin, and S.P. Boyd. Enhancing sparsity by reweighted $\ell_1$ minimization. *Journal of Fourier Analysis and Applications*, 14:877–905, 2008.

